# Joint MRI Bias Removal Using Entropy Minimization Across Images

**Erik G. Learned-Miller**
Department of Computer Science
University of Massachusetts, Amherst
Amherst, MA 01003

**Parvez Ahammad**
Division of Electrical Engineering
University of California, Berkeley
Berkeley, CA 94720

## Abstract

The correction of bias in magnetic resonance images is an important problem in medical image processing. Most previous approaches have used a maximum likelihood method to increase the likelihood of the pixels in a single image by adaptively estimating a correction to the unknown image bias field. The pixel likelihoods are defined either in terms of a pre-existing tissue model, or non-parametrically in terms of the image's own pixel values. In both cases, the specific location of a pixel in the image is not used to calculate the likelihoods. We suggest a new approach in which we simultaneously eliminate the bias from a set of images of the same anatomy, but from different patients. We use the statistics from the same location across different images, rather than within an image, to eliminate bias fields from all of the images simultaneously. The method builds a "multi-resolution" non-parametric tissue model conditioned on image location while eliminating the bias fields associated with the original image set. We present experiments on both synthetic and real MR data sets, and present comparisons with other methods.

## 1 Introduction

The problem of bias fields in magnetic resonance (MR) images is an important problem in medical imaging. This problem is illustrated in Figure 1. When a patient is imaged in the MR scanner, the goal is to obtain an image which is a function solely of the underlying tissue (left of Figure 1). However, typically the desired anatomical image is corrupted by a multiplicative bias field (2nd image of Figure 1) that is caused by engineering issues such as imperfections in the radio frequency coils used to record the MR signal. The result is a corrupted image (3rd image of Figure 1). (See [1] for background information.) The goal of MR bias correction is to estimate the uncorrupted image from the corrupted image.

A variety of statistical methods have been proposed to address this problem. Wells et al. [7] developed a statistical model using a discrete set of tissues, with the brightness distribution for each tissue type (in a bias-free image) represented by a one-dimensional Guassian distribution. An expectation-maximization (EM) procedure was then used to simultaneouly estimate the bias field, the tissue type, and the residual noise. While this method works well in many cases, it has several drawbacks: (1) Models must be developed *a priori* for each type of acquistion (for each different setting of the MR scanner), for each

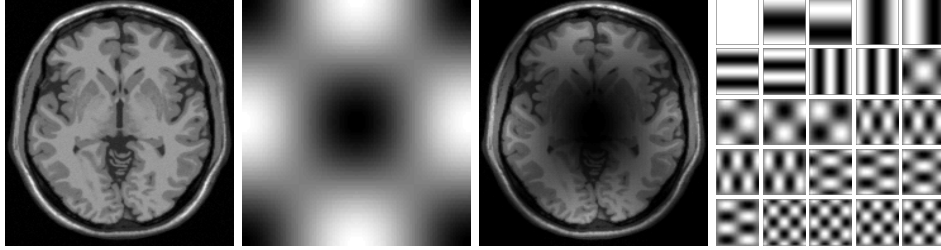

Figure 1: On the left is an idealized mid-axial MR image of the human brain with little or no bias field. The second image is a simulated low-frequency bias field. It has been exaggerated for ease of viewing. The third image is the result of pixelwise multiplication of the image by the bias field. The goal of MR bias correction is to recover the low-bias image on the left from the biased image on the right. On the right is the sine/cosine basis, used to construct band-limited bias fiels (see text).

new area of the body, and for different patient populations (like infants and adults). (2) Models must be developed from "bias-free" images, which may be difficult or impossible to obtain in many cases. (3) The model assumes a fixed number of tissues, which may be inaccurate. For example, during development of the human brain, there is continuous variability between gray matter and white matter. In addition, a discrete tissue model does not handle so-called partial volume effects in which a pixel represents a combination of several tissue types. This occurs frequently since many pixels occur at tissue boundaries.

Non-parametric approaches have also been suggested, as for example by Viola [10]. In that work, a non-parametric model of the tissue was developed from a single image. Using the observation that the entropy of the pixel brightness distribution for a *single image* is likely to increase when a bias field is added, Viola's method postulates a bias-correction field by minimizing the entropy of the resulting pixel brightness distribution. This approach addresses several of the problems of fixed-tissue parametric models, but has its own draw-backs: (1) The statistical model may be weak, since it is based on data from only a single image. (2) There is no mechanism for distinguishing between certain low-frequency image components and a bias field. That is, the method may mistake signal for noise in certain cases when removal of the true signal reduces the entropy of the brightness distriibution. We shall show that this is a problem in real medical images.

The method we present overcomes or improves upon problems associated with both of these methods and their many variations (see, e.g., [1] for recent techniques). It models tissue brightness non-parametrically, but uses data from multiple images to provide improved distribution estimates and alleviate the need for bias-free images for making a model. It also conditions on spatial location, taking advantage of a rich information source ignored in other methods. Experimental results demonstrate the effectiveness of our method.

## 2 The Image Model and Problem Formulation

We assume we are given a set $\mathbf{I}$ of observed images $I_i$ with $1 \leq i \leq N$, as shown on the left side of Figure 2. Each of these images is assumed to be the product of some bias-free image $L_i$ and a smooth bias field $B_i \in \mathcal{B}$. We shall refer to the bias-free images as *latent images* (also called *intrinsic images* by some authors). The set of all latent images shall be denoted $\mathbf{L}$ and the set of unknown bias fields $\mathbf{B}$. Then each observed image can be written as the product $I_i(x,y) = L_i(x,y) * B_i(x,y)$, where $(x,y)$ gives the pixel coordinates of each point, with $P$ pixels per image.

Consider again Figure 2. A *pixel-stack* through each image set is shown as the set of pixels corresponding to a particular location in each image (not necessarily the same tissue type). Our method relies on the principle that the pixel-stack values will have lower entropy when the bias fields have been removed. Figure 3 shows the simulated effect, on the distribution of values in a pixel-stack, of adding different bias fields to each image.

The latent image generation model assumes that each pixel is drawn from a fixed distribution $p_{x,y}(\cdot)$ which gives the probability of each gray value at the the location $(x,y)$ in the image. Furthermore, we assume that all pixels in the latent image are independent, given the distributions from which they are drawn. It is also assumed that the bias fields for each image are chosen independently from some fixed distribution over bias fields. Unlike most models for this problem which rely on statistical regularities within an image, we take a completely orthogonal approach by assuming that pixel values are independent given their image locations, but that pixel-stacks in general have low entropy when bias fields are removed.

We formulate the problem as a maximum a posteriori (MAP) problem, searching for the most probable bias fields given the set of observed images. Letting $\mathcal{B}$ represent the 25-dimensional product space of smooth bias fields (corresponding to the 25 basis images of Figure 1), we wish to find

$$\arg\max_{\mathbf{B}\in\mathcal{B}}P(\mathbf{B}|\mathbf{I}) \overset{(a)}{=} \arg\max_{\mathbf{B}\in\mathcal{B}}P(\mathbf{I}|\mathbf{B})P(\mathbf{B}) \tag{1}$$

$$\overset{(b)}{=} \arg\max_{\mathbf{B}\in\mathcal{B}}P(\mathbf{I}|\mathbf{B}) \tag{2}$$

$$\overset{(c)}{=} \arg\max_{\mathbf{B}\in\mathcal{B}}P(\mathbf{L}(\mathbf{I},\mathbf{B})) \tag{3}$$

$$= \arg\max_{\mathbf{B}\in\mathcal{B}}\prod_{x,y}\prod_{i=1}^{N}p_{x,y}(L_i(x,y)) \tag{4}$$

$$= \arg\max_{\mathbf{B}\in\mathcal{B}}\sum_{x,y}\sum_{i=1}^{N}\log p_{x,y}(L_i(x,y)) \tag{5}$$

$$\overset{(d)}{\approx} \arg\min_{\mathbf{B}\in\mathcal{B}}\sum_{x,y}H(p_{x,y}) \tag{6}$$

$$\overset{(e)}{\approx} \arg\min_{\mathbf{B}\in\mathcal{B}}\sum_{x,y}\hat{H}_{\text{Vasicek}}(L_1(x,y),...,L_N(x,y)) \tag{7}$$

$$= \arg\min_{\mathbf{B}\in\mathcal{B}}\sum_{x,y}\hat{H}_{\text{Vasicek}}(\frac{I_1(x,y)}{B_1(x,y)},...,\frac{I_N(x,y)}{B_N(x,y)}). \tag{8}$$

Here $H$ is the Shannon entropy $(-E(\log P(x)))$ and $\hat{H}_{\text{Vasicek}}$ is a sample-based entropy estimator.[1] (a) is just an application of Bayes rule. (b) assumes a uniform prior over the allowed bias fields. The method can easily be altered to incorporate a non-uniform prior.

$$\hat{H}_{\text{Vasicek}}(Z^1,...,Z^N) = \frac{1}{N-m}\sum_{i=1}^{N-m}\log\left(\frac{N}{m}(Z^{(i+m)}-Z^{(i)})\right), \tag{9}$$

where $Z^i$'s represent the values in a pixel-stack, $Z^{(i)}$'s represent those same values in rank order, $N$ is the number of values in the pixel-stack and $m$ is a function of $N$ (like $N^{0.5}$) such that $m/N$ goes to 0 as $m$ and $N$ go to infinity. These entropy estimators are discussed at length elsewhere [3].

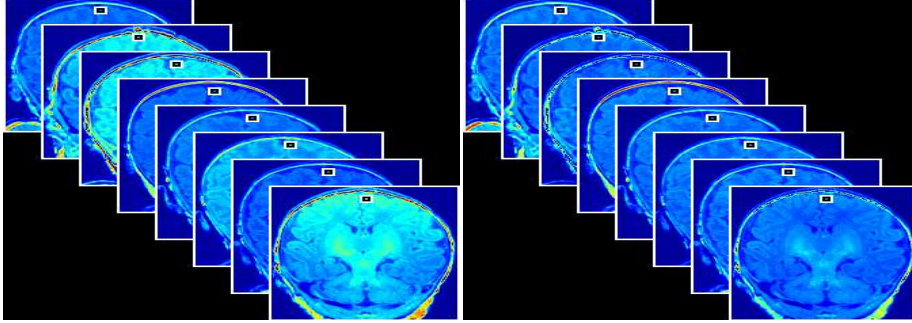

Figure 2: **On the left** are a set of mid-coronal brain images from eight different infants, showing clear signs of bias fields. A *pixel-stack*, a collection of pixels at the same point in each image, is represented by the small square near the top of each image. Although there are probably no more than two or three tissue types represented by the pixel-stack, the brightness distribution through the pixel-stack has high empirical entropy due to the presence of different bias fields in each image. **On the right** are a set of images that have been corrected using our bias field removal algorithm. While the images are still far from identical, the pixel-stack entropies have been reduced by mapping similar tissues to similar values in an "unsupervised" fashion, i.e. without knowing or estimating the tissue types.

(c) expresses the fact that the probability of the observed image given a particular bias field is the same as the probability of the latent image associated with that observed image and bias field. The approximation (d) replaces the empirical mean of the log probability at each pixel with the negative entropy of the underlying distribution at that pixel. This entropy is in turn estimated (e) using the entropy estimator of Vasicek [6] directly from the samples in the pixel-stack, without ever estimating the distributions $p_{x,y}$ explicitly. The inequality (d) becomes an equality as $N$ grows large by the law of large numbers, while the consistency of Vasicek's entropy estimator [2] implies that (e) also goes to equality with large $N$. (See [2] for a review of entropy estimators.)

## 3   The Algorithm

Using these ideas, it is straightforward to construct algorithms for joint bias field removal. As mentioned above, we chose to optimize Equation (8) over the set of band-limited bias fields. To do this, we parameterize the set of bias fields using the sine/cosine basis images shown on the right of Figure 1:

$$B_i = \sum_{j=1}^{25} \alpha_j \phi_j(x,y).$$

We optimize Equation (8) by *simultaneously* updating the bias field estimates (taking a step along the numerical gradient) for each image to reduce the overall entropy. That is, at time step $t$, the coefficients $\alpha_j$ for each bias field are updated using the latent image estimates and entropy estimates from time step $t-1$. After all $\alpha$'s have been updated, a new set of latent images and pixel-stack entropies are calculated, and another gradient step is taken. Though it is possible to do a full gradient descent to convergence by optimizing one image at a time, the optimization landscape tends to have more local minima for the last few images in the process. The appeal of our joint gradient descent method, on the other hand, is that the ensemble of images provides a natural smoothing of the optimization landscape

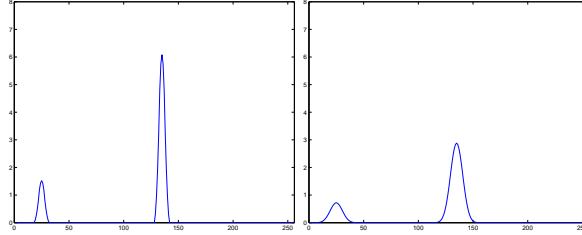

Figure 3: **On the left** is a simulated distribution from a pixel-stack taken through a particular set of bias-free mid-axial MR images. The two sharp peaks in the brightness distribution represent two tissues which are commonly found at that particular pixel location. **On the right** is the result of adding an independent bias field to each image. In particular, the spread, or entropy, of the pixel distribution increases. In this work, we seek to remove bias fields by seeking to reduce the entropy of the pixel-stack distribution to its original state.

in the joint process. It is in this sense that our method is "multi-resolution", proceeding from a smooth optimization in the beginning to a sharper one near the end of the process.

We now summarize the algorithm:

1. Initialize the bias field coefficients for each image to 0, with the exception of the coefficient for the DC-offset (the constant bias field component), which is initialized to 1. Initialize the gradient descent step size $\delta$ to some value.

2. Compute the summed pixelwise entropies for the set of images with initial "neutral" bias field corrections. (See below for method of computation.)

3. Iterate the following loop until no further changes occur in the images.

    (a) For each image:
        i. Calculate the numerical gradient $\nabla_\alpha H_{\text{Vasicek}}$ of (8) with respect to the bias field coefficients ($\alpha_j$'s) for the current image.
        ii. Set $\alpha = \alpha + \delta \nabla_\alpha \hat{H}_{\text{Vasicek}}$.
    (b) Update $\delta$ (reduce its value according to some schedule).

Upon convergence, it is assumed that the entropy has been reduced as much as possible by changing the bias fields, unless one or more of the gradient descents is stuck in a local minimum. Empirically, the likelihood of sticking in local minima is dramatically reduced by increasing the number of images ($N$) in the optimization. In our experiments described below with only 21 real infant brains, the algorithm appears to have found a global minimum of all bias fields, at least to the extent that this can be discerned visually.

Note that for a set of *identical* images, the pixel-stack entropies are not increased by multiplying each image by the same bias field (since all images will still be the same). More generally, when images are approximately equivalent, their pixel-stack entropies are not signficantly affected by a "common" bias field, i.e. one that occurs in all of the images.[2] This means that the algorithm cannot, in general, eliminate all bias fields from a set of images, but can only *set all of the bias fields to be equivalent.* We refer to any constant bias field remaining in all of the images after convergence as the *residual bias field*.

Fortunately, there is an effect that tends to minimize the impact of the residual bias field in many test cases. In particular, the residual bias field tends to consist of components for each $\alpha_j$ that approximate the mean of that component across images. For example, if half of the observed images have a positive value for a particular component's coefficient, and half have a negative coefficient for that component, the residual bias field will tend to have a coefficient near zero for that component. Hence, the algorithm naturally eliminates bias field effects that are non-systematic, i.e. that are not shared across images.

If the same type of bias field component occurs in a majority of the images, then the algorithm will not remove it, as the component is indistinguishable, under our model, from the underlying anatomy. In such a case, one could resort to within-image methods to further reduce the entropy. However, there is a risk that such methods will remove components that actually represent smooth gradations in the anatomy. This can be seen in the bottom third of Figure 4, and will be discussed in more detail below.

## 4  Experiments

To test our algorithm, we ran two sets of experiments, the first on synthetic images for validation, and the second on real brain images. We obtained synthetic brain images from the BrainWeb project [8, 9] such as the one shown on the left of Figure 1. These images can be considered "idealized" MR images in the sense that the brightness values for each tissue are constant (up to a small amount of manually added isotropic noise). That is, they contain no bias fields. The initial goal was to ensure that our algorithm could remove synthetically added bias fields, in which the bias field coefficients were known. Using $K$ copies of a single "latent" image, we added known but different bias fields to each one. For as few as five images, we could reliably recover the known bias field coefficients, up to a fixed offset for each image, to within 1% of the power of the original bias coefficients.

More interesting are the results on real images, in which the latent images come from different patients. We obtained 21 pre-registered[3] infant brain images (top of Figure 4) from Brigham and Women's Hospital in Boston, Massachusetts. Large bias fields can be seen in many of the images. Probably the most striking is a "ramp-like" bias field in the sixth image of the second row. (The top of the brain is too bright, while the bottom is too dark.) Because the brain's white matter is not fully developed in these infant scans, it is difficult to categorize tissues into a fixed number of classes as is typically done for adult brain images; hence, these images are not amenable to methods based on specific tissue models developed for adults (e.g. [7]).

The middle third of Figure 4 shows the results of our algorithm on the infant brain images. (These results must be viewed in color on a good monitor to fully appreciate the results.) While a trained technician can see small imperfections in these images, the results are remarkably good. All major bias artifacts have been removed.

It is interesting to compare these results to a method that reduces the entropy of each image individually, without using constraints between images. Using the results of our algorithm as a starting point, we continued to reduce the entropy of the pixels *within* each image (using a method akin to Viola's [10]), rather than across images. These results are shown in the bottom third of Figure 4. Carefully comparing the central brain regions in the middle section of the figure and the bottom section of the figure, one can see that the butterfly shaped region in the middle of the brain, which represents developing white matter, has

been suppressed in the lower images. This is most likely because the entropy of the pixels *within a particular image* can be reduced by increasing the bias field "correction" in the central part of the image. In other words, the algorithm strives to make the image more uniform by removing the bright part in the middle of the image. However, our algorithm, which compares pixels across images, does not suppress these real structures, since they occur across images. Hence coupling across images can produce superior results.

## 5  Discussion

The idea of minimizing pixelwise entropies to remove nuisance variables from a set of images is not new. In particular, Miller et al. [4, 5] presented an approach they call *congealing* in which the sum of pixelwise entropies is minimized by *separate affine transforms* applied to each image. Our method can thus be considered an extension of the congealing process to non-spatial transformations. Combining such approaches to do registration and bias removal simultaneously, or registration and lighting rectification of faces, for example, is an obvious direction for future work.

This work uses information unused in other methods, i.e. information across images. This suggests an iterative scheme in which both types of information, both within and across images, are used. Local models could be based on weighted neighborhoods of pixels, *pixel cylinders*, rather than single pixel-stacks, in sparse data scenarios. For "easy" bias correction problems, such an approach may be overkill, but for difficult problems in bias correction, where the bias field is difficult to separate from the underlying tissue, as discussed in [1], such an approach could produce critical extra leverage.

We would like to thank Dr. Terrie Inder and Dr. Simon Warfield for graciously providing the infant brain images for this work. The images were obtained under NIH grant P41 RR13218. Also, we thank Neil Weisenfeld and Sandy Wells for helpful discussions. This work was partially supported by Army Research Office grant DAAD 19-02-1-0383.

## Footnotes

[1]The entropy estimator used is similar to Vasicek's estimator [6], given (up to minor details) by

[2]Actually, multiplying each image by a bias field of small magnitude can artificially reduce the entropy of a pixel-stack, but this is only the result of the brightness values shrinking towards zero. Such artificial reductions in entropy can be avoided by normalizing a distribution to unit variance between iterations of computing its entropy, as is done in this work.

[3]It is interesting to note that registration is not strictly necessary for this algorithm to work. The proposed MAP method works under very broad conditions, the main condition being that the bias fields do not span the same space as parts of the actual medical images. It is true, however, that as the latent images become less registered or differ in other ways, that a much larger number of images is needed to get good estimates of the pixel-stack distributions.

## References

[1] Fan, A., Wells, W., Fisher, J., Cetin, M., Haker, S., Mulkern, C., Tempany, C., Willsky, A. A unified variational approach to denoising and bias correction in MR. Proceedings of IPMI, 2003.

[2] Beirlant, J., Dudewicz, E., Gyorfi, L. and van der Meulen, E. Nonparametric entropy estimation: An overview. *International Journal of Mathematical and Statistical Sciences, 6*. pp.17-39. 1997.

[3] Learned-Miller, E. G. and Fisher, J. ICA using spacings estimates of entropy. *Journal of Machine Learning Research*, Volume 4, pp. 1271-1295, 2003.

[4] Miller, E. G., Matsakis, N., Viola, P. A. Learning from one example through shared densities on transforms. *IEEE Conference on Computer Vision and Pattern Recognition*. 2000.

[5] Miller, E. G. Learning from one example in machine vision by sharing probability densities. Ph.D. thesis. Massachusetts Institute of Technology. 2002.

[6] Vasicek, O. A test for normality based on sample entropy. *Journal of the Royal Statistical Society Series B, 31*. pp. 632-636, 1976.

[7] Wells, W. M., Grimson, W. E. L., Kikinis, R., Jolesz, F. Adaptive segmentation of MRI data. *IEEE Transactions on Medical Imaging, 15*. pp. 429-442, 1996.

[8] Collins, D.L., Zijdenbos, A.P., Kollokian, J.G., Sled, N.J., Kabani, C.J., Holmes, C.J., Evans, A.C. Design and Construction of a realistic digital brain phantom. IEEE Transactions on Medical Imaging, 17. pp. 463-468, 1998.

[9] http://www.bic.mni.mcgill.ca/brainweb/

[10] Viola, P.A. Alignment by maximization of mutual information. Ph.D. Thesis. Massachusetts Institute of Technology. 1995.

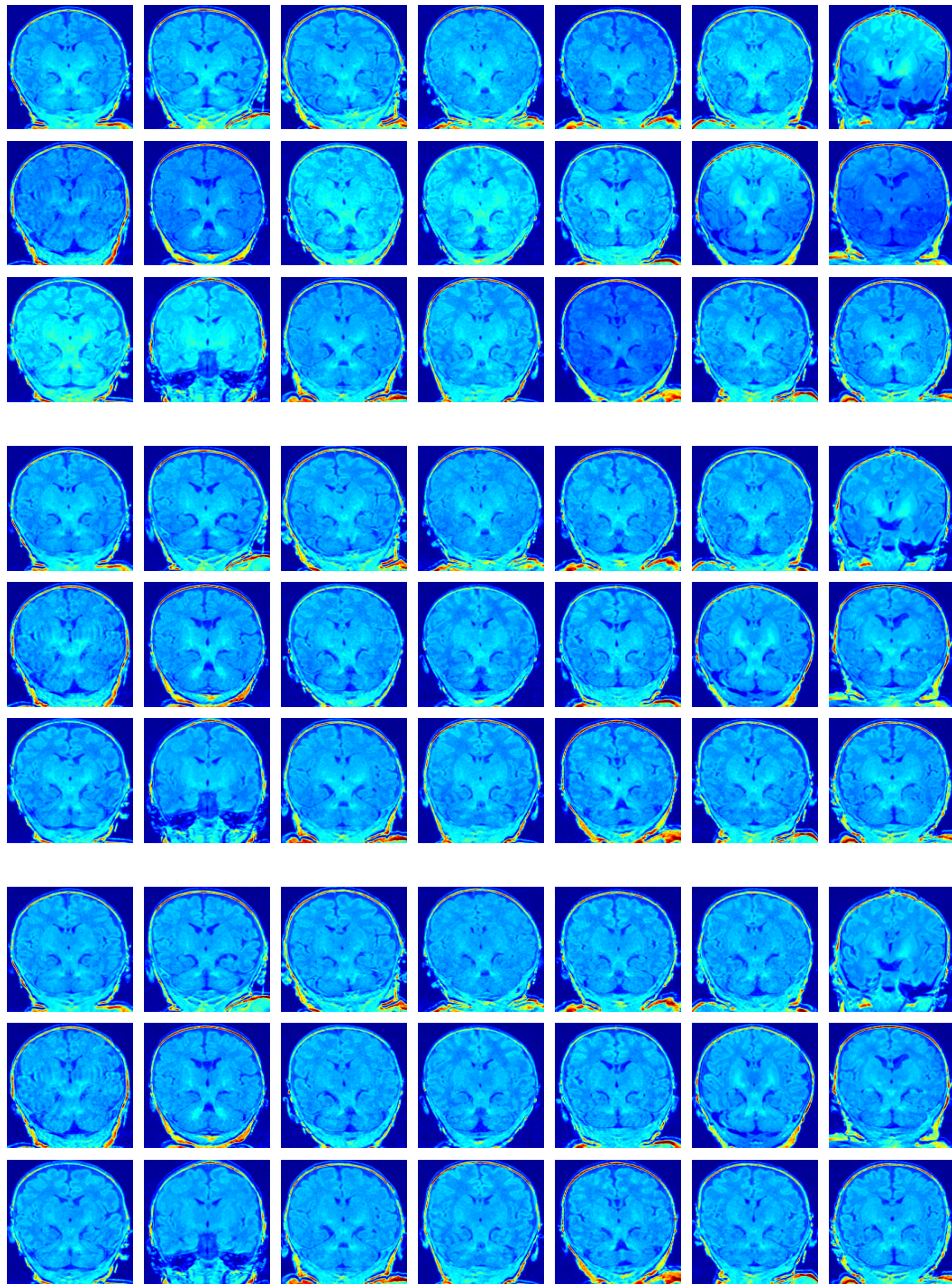

Figure 4: **NOTE**: This image must be viewed in color (preferably on a bright display) for full effect. **Top.** Original infant brain images. **Middle.** The same images after bias removal with our algorithm. Note that developing white matter (butterfly-like structures in middle brain) is well-preserved. **Bottom.** Bias removal using a single image based algorithm. Notice that white matter structures are repressed.